# An urn model for majority voting in classification ensembles

**Victor Soto**
Computer Science Department
Columbia University
New York, NY, USA
vsoto@cs.columbia.edu

**Alberto Suárez and Gonzalo Martínez-Muñoz**
Computer Science Department
Universidad Autónoma de Madrid
Madrid, Spain
{gonzalo.martinez,alberto.suarez}@uam.es

## Abstract

In this work we analyze the class prediction of parallel randomized ensembles by majority voting as an urn model. For a given test instance, the ensemble can be viewed as an urn of marbles of different colors. A marble represents an individual classifier. Its color represents the class label prediction of the corresponding classifier. The sequential querying of classifiers in the ensemble can be seen as draws without replacement from the urn. An analysis of this classical urn model based on the hypergeometric distribution makes it possible to estimate the confidence on the outcome of majority voting when only a fraction of the individual predictions is known. These estimates can be used to speed up the prediction by the ensemble. Specifically, the aggregation of votes can be halted when the confidence in the final prediction is sufficiently high. If one assumes a uniform prior for the distribution of possible votes the analysis is shown to be equivalent to a previous one based on Dirichlet distributions. The advantage of the current approach is that prior knowledge on the possible vote outcomes can be readily incorporated in a Bayesian framework. We show how incorporating this type of problem-specific knowledge into the statistical analysis of majority voting leads to faster classification by the ensemble and allows us to estimate the expected average speed-up beforehand.

## 1 Introduction

Combining the outputs of multiple predictors is in many cases of interest a successful strategy to improve the capabilities of artificial intelligence systems, ranging from agent architectures [19], to committee learning [13, 15, 8, 9]. A common approach is to build a collection of individual subsystems and then integrate their outputs into a final decision by means of a voting process. Specifically, in the machine learning literature, there is extensive empirical evidence on the improvements in generalization capacity that can be obtained using ensembles of learners [7, 11]. However, one of the drawbacks of these types of systems is the linear memory and time costs incurred in the computation of the final ensemble prediction by combination of the individual predictions. There are various strategies that alleviate these shortcomings. These techniques are grouped into static (or off-line) and dynamic (or online). In static pruning techniques, only a subset of complementary predictors from the original ensemble is kept [16, 21, 6]. By contrast, in dynamic pruning, the whole ensemble is retained. The prediction of the class label of a particular instance is accelerated by halting the sequential querying process when it is unlikely that the remaining (unknown) votes would change the output prediction [10, 20, 14, 12, 2, 3, 17]. These techniques are online in the sense that, as new individual predictions become known, the algorithm dynamically updated the estimated probability of having a stable prediction; i.d. a prediction that coincides with that by the complete ensemble. This is the basis of the *Statistical Instance-Based Algorithm* (SIBA) proposed in [14]. In a similar

approach, albeit with a different objective, Reyzin proposes to randomly sample hypotheses from the original AdaBoost ensemble. The goal is to minimize the number of features that are used for prediction, with a limited loss of accuracy [18]. This feature-efficient prediction is beneficial when access to the features of a new instance at test time is costly (e.g., in some medical problems). A different approach is followed in [3]. In this work, a policy is learned to decide which classifiers should be queried and which discarded in the prediction of the class label of a given instance.

The dynamic ensemble pruning method proposed in this work is closely related to SIBA [14]. In SIBA, the members of a committee are queried sequentially. At each step in the querying process, the votes recorded are used to estimate the probability that the majority decision of the classifiers queried up to that moment coincides with the complete ensemble. If this probability exceeds a specified confidence level, $\alpha$, the voting process is halted. To compute this estimate, the probability that a single predictor outputs a given decision for the particular instance considered is modeled as a random variable. Starting from a uniform prior, Bayes' theorem is used to update the distribution of this variable with the information provided by the actual votes, as they become known. In most of the problems analyzed in [14], the assumption that the prior is uniform leads to conservative estimates of the confidence on the stability of the predictions when only a fraction of the classifiers have been queried. Analyzing the results of those experiments, it is apparent that the actual disagreement percentages between the dynamic decision output and the decision made by the complete committee are significantly lower than the specified target $\alpha$. As a consequence, more queries are made than the ones that are actually needed.

The present work has two objectives. First, we propose an intuitive mathematical modeling of the voting process in ensembles of classifiers based on the hypergeometric distribution. Under the assumption that the distribution of possible vote outcomes is uniform, we prove that this derivation is equivalent to the one presented in [14]. However, the vote distribution is, in general, not uniform. Its shape depends on the classification task considered and on the base learning algorithm used to generate the predictors. Second, to take into account this dependence, we propose to approximate this distribution using a non-parametric prior. The use of this problem-specific prior knowledge leads to more accurate estimations of the disagreement rates between the dynamic sub-committee prediction and the complete committee, which are closer to the specified target $\alpha$. In this manner, faster classification can be achieved with minimal loss of accuracy. In addition, the use of priors allow us to estimate quite precisely the expected average number of trees that would be necessary to query.

## 2 Modeling ensemble voting processes as a classical urn problem

Consider the following process modeled as a classical urn model. Let us suppose we have marbles of $l$ different colors in an urn. The number of marbles of color $y_k$ in the urn is $T_k$, with $k = 1 \ldots l$. The total number of marbles in the urn is $T = \sum_{k=1}^{l} T_k$. The contents of the urn can therefore be described by vector $\mathbf{T} = \langle T_1, T_2 \ldots T_l \rangle$. Assume that $t < T$ marbles are extracted from the urn without replacement. This extraction process can be characterized by vector $\mathbf{t} = \langle t_1, t_2 \ldots t_l \rangle$ where $t_k$ is the number of marbles of color $y_k$ extracted, with $t = \sum_{k=1}^{l} t_k$. The probability of extracting a color distribution of marbles $\mathbf{t}$, given the initial color distribution of the urn $\mathbf{T}$ is described by the multivariate hypergeometric distribution

$$\mathcal{P}(\mathbf{t}|\mathbf{T}) = \frac{\binom{T_1}{t_1} \cdots \binom{T_l}{t_l}}{\binom{T}{t}} = \frac{\prod_{i=1}^{l} \binom{T_i}{t_i}}{\binom{T}{t}} \ . \tag{1}$$

Consider the case in which the total number of marbles in the urn, $T$, is known but that the color distribution, $\mathbf{T}$, is unknown. In this case, the color distribution of the extracted marbles, $\mathbf{t}$, can be used to estimate the content of the urn applying Bayes Theorem

$$\mathcal{P}(\mathbf{T}|\mathbf{t}) = \frac{\mathcal{P}(\mathbf{t}|\mathbf{T})\mathcal{P}(\mathbf{T})}{\mathcal{P}(\mathbf{t})} = \frac{\mathcal{P}(\mathbf{t}|\mathbf{T})\mathcal{P}(\mathbf{T})}{\sum_{\mathbf{T}^* \in \Omega_{\mathbf{t}}} \mathcal{P}(\mathbf{t}|\mathbf{T}^*)\mathcal{P}(\mathbf{T}^*)} = \frac{\binom{T_1}{t_1} \cdots \binom{T_l}{t_l}\mathcal{P}(\mathbf{T})}{\sum_{\mathbf{T}^* \in \Omega_{\mathbf{t}}} \binom{T_1^*}{t_1} \cdots \binom{T_l^*}{t_l}\mathcal{P}(\mathbf{T}^*)} \tag{2}$$

where $\Omega_{\mathbf{t}}$ is the set of vectors $\mathbf{T}^*$, such that $T_i^* \geq t_i \ \forall i$ and $\sum_{i=1}^{l} T_i^* = T$.

This problem is equivalent to the voting process in an ensemble of classifiers: Suppose we want to predict the class label of an instance by combining the individual predictions of the ensemble classifiers (marbles). Assuming that the individual predictions are deterministic, the class (color) that

each classifier (marble) would output if queried is fixed, but unknown before the query. Therefore, for each instance considered we have a different "bag of colored marbles" with an unknown class distribution. After a partial count of votes of the ensemble is known, Eq. 2 provides an estimate of the distribution of votes for the complete ensemble. This estimate can be used to compute the probability that the decision obtained using only a partial tally of votes, $\mathbf{t}$, of size $t < T$ and by the final decision using all $T$ votes, coincide

$$\mathcal{P}^*(\mathbf{t}, T) = \sum_{\mathbf{T} \in \mathcal{T}_\mathbf{t}} \frac{\binom{T_1}{t_1} \dots \binom{T_l}{t_l} \mathcal{P}(\mathbf{T})}{\sum_{\mathbf{T}^* \in \Omega_\mathbf{t}} \binom{T_1^*}{t_1} \dots \binom{T_l^*}{t_l} \mathcal{P}(\mathbf{T}^*)}, \tag{3}$$

where $\mathcal{T}_\mathbf{t}$ is the set of vectors of votes for the complete ensemble $\mathbf{T} = \{T_1, T_2 \dots T_l\}$ such that the class predicted by the subensemble of size $t$ and the class predicted by the complete committee coincide, with $T_i \geq t_i$, and $\sum_{i=1}^l T_i = T$.

If $\mathcal{P}^*(\mathbf{t}, T) = 1$, then the classification given by the partial ensemble and the full ensemble coincide. This case happens when the difference between the number of votes for the first and second class in $\mathbf{t}$ is greater than the remaining votes in the urn. In such case, the voting process can be halted with full confidence that the decision of the partial ensemble will not change when the predictions of the remaining classifiers are considered. In addition, if it is acceptable that, with a small probability $1 - \alpha$, the prediction of the partially polled ensemble and that of the complete ensemble disagree, then the voting process can be stopped when the $\mathcal{P}^*(\mathbf{t}, T)$ exceeds the specified confidence level $\alpha$. The final classification would be given as the combined decisions of the classifiers that have been polled up to that point only.

## 2.1 Uniform prior

Assuming a uniform prior for the distribution of possible $\mathbf{T}$ vectors $\mathcal{P}(\mathbf{T}) = 1/\|\mathbf{T}\|$, where $\|\mathbf{T}\|$ stands for the number of possible $\mathbf{T}$ vectors, this derivation is equivalent to the one presented in [14]. That formulation assumes that the base classifiers of the ensemble are independent realizations from a pool of all possible classifiers given the training dataset. Assuming that an unlimited number of realizations can be performed, the distribution of class votes in the ensemble converges to a Dirichlet distribution in the limit of infinite ensemble size. Then, assuming a partial tally of $t$ votes, the probability that the ensemble's decision will change if the precictions of the remaining $T - t$ classifiers are considered, can be estimated.

In order to prove the equivalence between both formulations, we first need to introduce three results, presented in the theorem and propositions below.

**Theorem.** Chu-Vandermonde Identity. Let $s, t, r \in \mathbb{N}$ then

$$\binom{s+t}{r} = \sum_{k=0}^{r} \binom{s}{k} \binom{t}{r-k} \tag{4}$$

**Proposition 1.** Upper negation. Let $r \in \mathbb{C}$ and $k \in \mathbb{Z}$, then

$$\binom{r}{k} = (-1)^k \binom{k-r-1}{k} \tag{5}$$

The previous theorem and proposition are used in the following proposition, which is the key to prove the equivalence between the two formulations:

**Proposition 2.** Let $n_1$ and $n_2$ be positive integers such that $n_1 + n_2 = n$ and $n \leq N$. Then

$$\sum_{i=n_1}^{N-n_2} \binom{i}{n_1} \binom{N-i}{n_2} = \binom{N+1}{N-n} \tag{6}$$

*Proof.* First the symmetry property of the binomial (i.e., $\binom{n}{k} = \binom{n}{n-k}$) is used to bring down the indices

$$\sum_{i=n_1}^{N-n_2} \binom{i}{n_1} \binom{N-i}{n_2} = \sum_{i=n_1}^{N-n_2} \binom{i}{i-n_1} \binom{N-i}{N-i-n_2}$$

The upper indices are removed by applying the upper negation property of proposition 1.

$$\sum_{i=n_1}^{N-n_2} \binom{i}{i-n_1}\binom{N-i}{N-i-n_2} = \sum_{i=n_1}^{N-n_2} \binom{-n_1-1}{i-n_1}\binom{-n_2-1}{N-i-n_2}(-1)^{i-n_1}(-1)^{N-i-n_2}$$

Now, the Chu-Vandermonde identity can be applied with $r = N - n_1 - n_2$ and $k = i - n_1$

$$\sum_{i=n_1}^{N-n_2} \binom{-n_1-1}{i-n_1}\binom{-n_2-1}{N-i-n_2}(-1)^{i-n_1}(-1)^{N-i-n_2} = \binom{-n-2}{N-n}(-1)^{N-n}$$

Finally the upper negation is applied again

$$\binom{-n-2}{N-n}(-1)^{N-n} = \binom{N+1}{N-n}$$

$\square$

**Proposition 3** Following the hypergeometric reformulation given by Equation 2 and assuming that $\mathcal{P}(\mathbf{T})$ follows a uniform distribution $1/\|\mathbf{T}\|$, where $\|\mathbf{T}\|$ stands for the number of possible $\mathbf{T}$ vectors then

$$\mathcal{P}(\mathbf{T}|\mathbf{t}) = \frac{(T-t)!}{\prod_{i=1}^{l}(T_i-t_i)!}\frac{\prod_{i=1}^{l}(t_i+1)_{T_i-t_i}}{(t+l)_{T-t}}$$

where $(x)_n = x(x+1)\ldots(x+n-1)$ is the Pochhammer symbol. This formulation is equivalent to the one proposed in [14].

*Proof.* Equation 2 can be simplified by taking into account the uniform prior $\mathcal{P}(\mathbf{T}) = 1/\|\mathbf{T}\|$ as

$$\mathcal{P}(\mathbf{T}|\mathbf{t}) = \frac{\mathcal{P}(\mathbf{t}|\mathbf{T})\mathcal{P}(\mathbf{T})}{\mathcal{P}(\mathbf{t})} = \frac{\binom{T_1}{t_1}\ldots\binom{T_l}{t_l}}{\sum_{\mathbf{T}^*\in\Omega_{\mathbf{t}}}\binom{T_1^*}{t_1}\ldots\binom{T_l^*}{t_l}} \tag{7}$$

The indices of the summation, $\Omega_{\mathbf{t}}$, is the set of vectors $\mathbf{T}$ such that $T_i \geq t_i\ \forall i$ and $\sum_{i=1}^{l}T_i = T$. They can be changed for $l$ classes to

$$\mathcal{P}(\mathbf{T}|\mathbf{t}) = \frac{\binom{T_1}{t_1}\ldots\binom{T_l}{t_l}}{\sum_{T_1^*=t_1}^{\hat{T}_1}\sum_{T_2^*=t_2}^{\hat{T}_2}\cdots\sum_{T_{l-1}^*=t_{l-1}}^{\hat{T}_{l-1}}\binom{T_1^*}{t_1}\ldots\binom{T_l^*}{t_l}} \tag{8}$$

where $\hat{T}_k$ for $k = 1,\ldots,(l-1)$ are the maximum values for $T_k^*$ in the summations. Note that the summation over $T_l^*$ is unnecessary since the value of $T_l^*$ becomes fixed once the values of $T_1^*\ldots T_{l-1}^*$ are fixed since $\sum_{i=1}^{l}T_i^* = T$. In this sense, the values for $\hat{T}_k$ have a dependency on $T_i^*$ for $i < k$ as $\hat{T}_k = T - t + t_k - \sum_{i=1}^{k-1}(T_i^* - t_i)$, $k = 1,\ldots,(l-1)$. The summations in the denominator of Eq. 8 can be rearranged

$$\sum_{T_1^*=t_1}^{\hat{T}_1}\sum_{T_2^*=t_2}^{\hat{T}_2}\cdots\sum_{T_{l-1}^*=t_{l-1}}^{\hat{T}_{l-1}}\binom{T_1^*}{t_1}\ldots\binom{T_l^*}{t_l} = \sum_{T_1^*=t_1}^{\hat{T}_1}\binom{T_1^*}{t_1}\sum_{T_2^*=t_2}^{\hat{T}_2}\binom{T_2^*}{t_2}\cdots\sum_{T_{l-1}^*=t_{l-1}}^{\hat{T}_{l-1}}\binom{T_{l-1}^*}{t_{l-1}}\binom{T_l^*}{t_l}.$$

Proposition 2 (Eq. 6) can be used, together with $N = T - \sum_{i=1}^{l-2}T_i^*$, to express the summation over $T_{l-1}^*$ in closed form

$$\sum_{T_{l-1}^*=t_{l-1}}^{T-t+t_{l-1}-\sum_{i=1}^{l-2}(T_i^*-t_i)}\binom{T_{l-1}^*}{t_{l-1}}\binom{T_l^*}{t_l} = \sum_{T_{l-1}^*=t_{l-1}}^{T-\sum_{i=1}^{l-2}T_i^*-t_l}\binom{T_{l-1}^*}{t_{l-1}}\binom{T-\sum_{i=1}^{l-2}T_i^*-T_{l-1}^*}{t_l} =$$

$$\binom{T-\sum_{i=1}^{l-2}T_i^*+1}{T-\sum_{i=1}^{l-2}T_i^*-t_{l-1}-t_l} = \binom{T-\sum_{i=1}^{l-2}T_i^*+1}{t_{l-1}+t_l+1},$$

where the symmetry property of the binomial has been used in the last step. The subsequent summations are carried out in the same manner. The summation over $T_k^*$ requires the application of Eq. 6 with $N = T - \sum_{i=1}^{k-1} T_i^* + (l - k - 1)$, $n_1 = t_k$ and $n_2 = \sum_{i=k+1}^{l} t_i + (l - k - 1)$

$$\sum_{T_1^*=t_1}^{\hat{T}_1} \binom{T_1^*}{t_1} \cdots \sum_{T_{l-2}^*=t_{l-2}}^{\hat{T}_{l-2}} \binom{T_{l-2}^*}{t_{l-2}} \binom{T - \sum_{i=1}^{l-2} T_i^* + 1}{t_{l-1} + t_l + 1} = \cdots = \binom{T + l - 1}{t + l - 1}$$

Employing this result in Eq. 8, one obtains

$$\mathcal{P}(\mathbf{T}|\mathbf{t}) = \frac{\binom{T_1}{t_1} \cdots \binom{T_l}{t_l}}{\binom{T+l-1}{t+l-1}} = \frac{\frac{T_1!}{t_1!(T_1-t_1)!} \cdots \frac{T_l!}{t_l!(T_l-t_l)!}}{\frac{(T+l-1)!}{(t+l-1)!(T-t)!}} = \frac{(T-t)!}{\prod_{i=1}^{l} (T_i - t_i)!} \frac{\prod_{i=1}^{l} (t_i + 1)_{T_i-t_i}}{(t+l)_{T-t}}.$$

$\square$

## 2.2  Non-uniform prior

The distribution $\mathcal{P}(\mathbf{T})$ can be modeled using a non-parametric non-uniform prior. The values of this prior can be obtained from the training data by some form of validation; e.g., out-of-bag or cross validation. Out-of-bag validation is faster because it does not require multiple generations of the ensemble. Therefore, it will be the validation method used in our implementation of the method. To compute the out-of-bag error, each training instance, $\mathbf{x}_n$, is classified by the ensemble predictors that do not have that particular instance in their training set. Let $\tilde{T}^n = \tilde{T}_1^n + \ldots + \tilde{T}_l^n$, be the number of such classifiers, where $\tilde{T}_i^n$ is the number of out-of-bag votes for class $i$, where $i = 1, \ldots, l$, assigned to instance $\mathbf{x}_n$. The number of votes for each class for an ensemble of size $T$ is estimated as $T_i^n \approx round(T \, \tilde{T}_i^n / \tilde{T}^n)$. To mitigate the influence of the random fluctuations that appear because of the finite size of the training set and to avoid spurious numeric artifacts, the prior is subsequently smoothed using a sliding window of size 5 over the vote distribution.

As shown in Section 2, the response time of the ensemble can be reduced by using Eq. 3, if we allow that a small fraction, $1 - \alpha$, of the predictions given by ensembles of size $\mathbf{t}$ and $\mathbf{T}$ do not coincide. Assuming this tolerance, when $\mathcal{P}^*(\mathbf{t}, T) > \alpha$, the voting process can be halted and the ensemble will output the decision given by the $t \leq T$ queried classifiers. However, the computation of Eq. 3 is costly and should be performed off-line. In the SIBA formulation, a lookup table indexed by the number of votes of the minority class (for binary problems) and whose values are the minimum number of votes of the majority class such that $\mathcal{P}^*(\mathbf{t}, T) > \alpha$, is used. Using a precomputed lookup table to halt the voting process does not entail a significant overhead during classification: a single lookup operation in the table is needed for each vote. The consequence of using a uniform prior is that all classes are considered equivalent. Hence, it is sufficient to compute one lookup table and use the minority class for indexing.

When prior knowledge is taken into account, the probability $\mathcal{P}^*(t_1 = n, t_2 = m, T)$ is not necessarily equal to $\mathcal{P}^*(t_1 = m, t_2 = n, T)$ for $n \neq m$. Therefore, a different lookup table per class will be necessary. In addition, it is necessary to compute a different set of tables for each dataset. In the original formulation, the lookup table values depend only on $T$ and $\alpha$. Therefore, they are independent of the particular classification problem considered. In our case, the prior distribution is estimated from the training data: Hence, it is problem dependent. However, the querying process is similar to SIBA. For instance, if we have 1 vote for class 1 and 7 for class 2, one determines whether the value in position 1 (minority class at this moment) of the lookup table for class 1 is greater or equal to 7. If it is, the querying process stops. As a side effect, for the experimental comparison, it is necessary to recompute the lookup tables for each realization of the data. Notwithstanding, in a real setting, these tables need to be computed only once. This can be done offline. Therefore, the speed improvements in the classification phase are independent of the size of the training set.

The lookup table and the estimated non-parametric prior can be used to estimate also the average number of classifiers that are expected to be queried during test. This estimation can be made using Monte Carlo simulation. To this end one would perform the following experiment repeatedly and compute the average number of queries: extract a random vector $\mathbf{T}$ from the prior distribution; generate a vector of votes of size $T$ that contains exactly $T_i$ votes for class i with $i = 1 \ldots l$; finally, query a random permutation of this vector of votes until the process can be halted as given by the lookup table and keep the number of queries.

# 3 Experiments

In this section we present the results of an extensive empirical evaluation of the dynamical ensemble pruning method described in the previous section. The experiments are performed in a series of benchmark classification problems from the UCI Repository [1] and synthetic data [4] using Random Forests [5]. The code is available at: https://github.com/vsoto/majority-ibp-prior.

The protocol for the experiments is as follows: for each problem, 100 partitions are created by $10 \times 10$-fold cross-validation for real datasets and by random sampling in the synthetic datasets. All the classification tasks considered are binary, except for *New-thyroid*, *Waveform* and *Wine*, which have three classes. For each partition, the following steps are carried out: (i) a Random Forest ensemble of size $T = 101$ is built; (ii) we compute the generalization error rate of the complete ensemble in the test set and record the mean number of trees that are queried to determine the final prediction. Note that this number need not be $T$: the voting process can be halted when the remaining votes (i.e. the predictions of classifiers that have not been queried up to that point) cannot modify the partial ensemble decision. This is the case when the number of remaining votes is below the difference between the majority class and the second most voted class; (iii) The SIBA algorithm [14] is applied to dynamically select the number of classifiers that are needed for each instance in the test set to achieve a level of confidence in the prediction above $\alpha = 0.99$. We use SIBA as the benchmark for comparison since in previous studies it has been shown to provide the best overall results, especially for $T < 500$ [2]; (iv) The process is repeated using the proposed method with non-uniform priors for the class vote distribution, with the same confidence threshold, $\alpha = 0.99$. The prior distribution $\mathcal{P}(\mathbf{T})$ is estimated in the training set using out-of-bag data. This prior is also used to estimate the expected number of trees to be queried in the testing phase. In addition, for steps (iii) and (iv) we compute the test error rate, the average number of queried trees, and the disagreement rates between the predictions of the partially queried ensembles and the complete ones.

Table 1: Error rates (left) and disagreement % (right). The statistical significant differences, using paired t-tests at a significance level $\alpha = 0.05$, are highlighted in boldface.

| Problem | Error rates | | | Disagreement % | |
|---|---|---|---|---|---|
| | RF | SIBA | HYPER | SIBA | HYPER |
| Australian | 13.00±3.7 | 13.09±3.7 | 13.25±3.8 | 0.3±0.6 | **0.9±1.1** |
| Breast | 3.22±2.1 | 3.23±2.1 | 3.76±2.3 | 0.1±0.4 | **1.0±1.1** |
| Diabetes | 24.34±4.2 | 24.25±4.1 | 24.23±4.0 | 0.6±0.9 | **0.8±1.0** |
| Echocardiogram | 22.18±14.3 | 22.05±14.7 | 22.18±14.1 | 0.7±3.1 | 1.4±4.6 |
| German | 23.43±3.5 | 23.65±3.3 | 23.62±3.3 | 0.8±0.8 | 0.8±0.9 |
| Heart | 18.30±6.9 | 18.37±7.0 | 18.37±7.2 | 0.8±1.8 | 1.0±2.1 |
| Horse-colic | 15.47±5.6 | 15.44±5.4 | 15.44±5.4 | 0.4±0.9 | **0.7±1.3** |
| Ionosphere | 6.44±4.1 | 6.44±4.1 | 6.52±3.9 | 0.1±0.6 | **0.7±1.3** |
| Labor | 6.33±8.9 | 6.17±8.8 | 6.43±9.1 | 0.2±1.7 | **1.2±4.5** |
| Liver | 27.10±6.7 | 27.09±7.0 | 27.01±6.9 | 1.0±1.7 | 0.9±1.5 |
| Mushroom | 0.00±0.0 | 0.00±0.0 | 0.08±0.2 | 0.0±0.0 | **0.1±0.2** |
| New-thyroid | 4.29±4.0 | 4.38±4.0 | 4.66±4.2 | 0.1±0.7 | **0.7±2.0** |
| Ringnorm | 7.60±1.3 | 7.72±1.2 | 7.82±1.2 | 0.5±0.2 | **0.8±0.3** |
| Sonar | 16.25±8.7 | 16.45±8.7 | 16.45±8.8 | 0.9±2.0 | 0.8±1.9 |
| Spam | 4.59±1.5 | 4.63±1.5 | 4.86±1.4 | 0.1±0.2 | **0.7±0.4** |
| Threenorm | 17.85±1.1 | 18.04±1.1 | 17.97±1.1 | 1.0±0.2 | **0.8±0.2** |
| Tic-tac-toe | 1.05±1.1 | 1.16±1.1 | 1.72±1.5 | 0.1±0.4 | **0.7±1.0** |
| Twonorm | 4.66±0.6 | 4.77±0.6 | 4.90±0.6 | 0.4±0.1 | **0.7±0.2** |
| Votes | 4.05±2.9 | 4.12±2.9 | 4.30±2.9 | 0.1±0.4 | **1.0±1.8** |
| Waveform | 17.30±0.9 | 17.36±0.8 | 17.45±0.8 | 0.6±0.1 | **1.0±0.3** |
| Wine | 1.69±2.8 | 1.74±2.8 | 2.30±3.5 | 0.1±0.6 | **1.1±2.5** |

In Table 1, we compare the error rates of Random Forest (RF) and of the dynamically pruned ensembles using the halting rule derived from assuming uniform priors (SIBA) and using non-uniform priors (HYPER), and the disagreement rates. The values displayed are averages over 100 realizations of the datasets The standard deviation is given after the ± symbol.

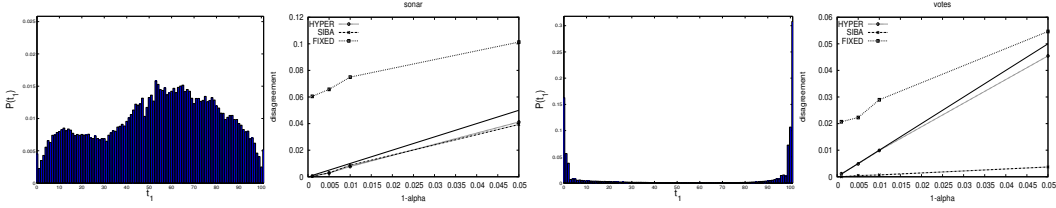

Figure 1: Vote distribution, $\mathcal{P}(\mathbf{T})$, and disagreement rates for *Sonar* (left) and *Votes* (right)

From Table 1, one observes that the mean error rates of the pruned ensembles using SIBA and HYPER are only slightly worse than the rates obtained by the complete ensemble (RF). These differences should be expected since we are allowing a small disagreement of $1 - \alpha = 1\%$ between the decisions of the partial and the complete ensemble. In any case, the differences in generalization error can be made arbitrarily small by increasing $\alpha$. By design, the disagreement rates are expected to be below, but close to $1\%$. From Table 1, one observes that the disagreement % of the proposed method (HYPER) are closer to the specified threshold ($1 - \alpha = 1\%$) than those of SIBA, except for *Liver*, *Sonar* and *Threenorm*, where the differences are small. In these problems (and in general in the problems where SIBA obtains disagreement rates closer to $1 - \alpha$), the distribution of $\mathbf{T}$ is closer to a uniform distribution (see Figure 1, left histogram). In consequence, the assumption of uniform prior taken by SIBA is closer to the real one. However, when $\mathcal{P}(\mathbf{T})$ differs from the uniform distribution (see for instance *Votes* in Figure 1 right histogram) the results of SIBA are rather different from the expected disagreement rates.

Table 2: Number of queried trees and speed-up rate with respect to the full ensemble of 101 trees. The statistical significant differences between SIBA and HYPER, using paired t-tests at a significance level $\alpha = 0.05$, are highlighted in boldface.

| Problem | # of trees | | | | Speed-up rate | | |
|---|---|---|---|---|---|---|---|
| | RF* | SIBA | HYPER | MC Estim | RF* | SIBA | HYPER |
| Australian | 62.2±1.4 | 16.1±2.1 | **12.8±2.3** | 12.9±0.9 | 1.6 | 6.3 | 7.9 |
| Breast | 54.2±0.9 | 8.9±1.4 | **4.0±1.0** | 4.0±0.4 | 1.9 | 11.3 | 25.3 |
| Diabetes | 68.8±1.8 | 24.9±3.2 | **24.0±3.2** | 23.8±1.1 | 1.5 | 4.1 | 4.2 |
| Echocardiogram | 68.0±4.6 | 22.6±8.2 | **20.0±8.0** | 21.6±3.2 | 1.5 | 4.5 | 5.1 |
| German | 71.8±1.3 | 28.4±2.8 | **27.7±2.9** | 30.1±1.0 | 1.4 | 3.6 | 3.6 |
| Heart | 67.2±2.5 | 22.5±4.2 | **20.9±4.2** | 20.7±1.7 | 1.5 | 4.5 | 4.8 |
| Horse-colic | 66.2±2.1 | 20.2±3.5 | **17.5±3.7** | 18.6±1.5 | 1.5 | 5.0 | 5.8 |
| Ionosphere | 57.9±1.5 | 11.9±2.3 | **7.8±2.1** | 7.8±0.6 | 1.7 | 8.5 | 12.9 |
| Labor | 61.6±4.0 | 14.1±6.0 | **9.7±5.3** | 10.2±2.0 | 1.6 | 7.2 | 10.4 |
| Liver | 74.5±2.3 | 31.8±4.5 | 31.7±4.5 | 31.6±2.0 | 1.4 | 3.2 | 3.2 |
| Mushroom | 51.0±0.0 | 6.0±0.0 | **1.0±0.0** | 1.0±0.0 | 2.0 | 16.8 | 101.0 |
| New-thyroid | 55.2±1.8 | 10.7±2.6 | **6.0±2.3** | 6.2±1.4 | 1.8 | 9.4 | 16.8 |
| Ringnorm | 68.6±0.8 | 22.9±1.1 | **20.4±1.5** | 19.7±2.3 | 1.5 | 4.4 | 5.0 |
| Sonar | 73.9±3.0 | **32.1±6.6** | 32.6±6.8 | 31.8±2.4 | 1.4 | 3.1 | 3.1 |
| Spam | 57.1±0.3 | 11.1±0.5 | **7.2±0.6** | 7.1±0.5 | 1.8 | 9.1 | 14.0 |
| Threenorm | 76.6±0.5 | **34.8±1.0** | 35.8±1.6 | 33.4±2.5 | 1.3 | 2.9 | 2.8 |
| Tic-tac-toe | 60.7±0.9 | 12.8±1.4 | **7.8±1.2** | 8.6±0.7 | 1.7 | 7.9 | 12.9 |
| Twonorm | 67.2±0.2 | 21.0±0.5 | **18.4±0.9** | 18.8±1.7 | 1.5 | 4.8 | 5.5 |
| Votes | 54.5±1.2 | 8.8±1.8 | **4.1±1.4** | 4.0±0.7 | 1.9 | 11.5 | 24.6 |
| Waveform | 72.3±0.7 | 29.3±1.1 | **27.8±1.7** | 28.6±2.8 | 1.4 | 3.4 | 3.6 |
| Wine | 57.3±2.1 | 11.4±2.7 | **5.8±1.8** | 6.7±1.4 | 1.8 | 8.9 | 17.5 |

In order to analyze this aspect in more detail, we have computed the disagreement rates for different values of alpha ($\alpha = 0.999, 0.995, 0.99, 0.95$). In Figure 1 the relation between the target $1 - \alpha$ and the actual disagreement rate is presented. A diagonal solid line marks the expected upper limit for the disagreement. The results for SIBA, HYPER and for the case of using a fixed number of trees for all instances (FIXED) (and equal to the average number of trees used by HYPER in those tasks)

are presented in these plots. This last case (FIXED) can be seen as a stochastic approximation to the prediction of the whole ensemble. From these plots, we observe that the results for HYPER are very close to the expected disagreement rates for cases in which the prior is approximately uniform (*Sonar*), and for cases in which the prior is non-uniform (*Votes*). As expected, the results of SIBA are close to the target only for the case of approximately uniform prior (*Sonar*). Finally, when a stochastic approximation is used (FIXED) the disagreement rates are clearly above the target threshold given by $\alpha$. From these results we conclude that the proposed model provides a more accurate description of the voting process used to compute the prediction of the ensemble. This means that taking into account the prior distribution of possible vote outcomes, $\mathcal{P}(\mathbf{T})$, is important to obtain disagreement rates that are closer to the threshold established.

Finally, in Table 2, we present the average number of trees used by Random Forest (RF*), the SIBA method, the proposed method using non-parametric priors (HYPER), and the expected average of the number of trees to be queried in HYPER using Monte Carlo sampling (MC Estim). Note that the number of trees used by RF* is not necessary $T = 101$: the voting process is halted when the remaining (unknown) predictions cannot alter the decision of the ensemble. The number of trees given in RF* is the same as the trees that SIBA or HYPER would use when $\alpha = 100\%$. Finally, the last three columns of Table 2 display the speed-up rate of the partial ensembles with respect to the full ensemble of size $T = 101$. From this table it is clear that HYPER reduces the number of queried classifiers with respect to SIBA in most of the tasks investigated. In addition, using only training data, the Monte Carlo estimations of the average number of trees are very precise. The largest average difference between this estimation and HYPER is $2.4$ trees for *German* and *Threenorm*. The speed-up rate of HYPER with respect to the full ensemble is remarkable: from $2.8$ times faster for *Threenorm* to 101 times faster in *Mushroom*. This dataset can be used to illustrate the benefits of using the prior distribution. For this problem, most classifiers agree in their predictions. HYPER takes advantage of this prior knowledge and queries only one classifier to cast the final decision. In this problem, the chances that the prediction of a single classifier, and the prediction of the complete ensemble are different, are below $1\%$. Similar behavior (but not as extreme) is observed in *Breast* and *Votes*.

## 4 Conclusions

In this work, we present an intuitive, rigorous mathematical description of the voting process in an ensemble of classifiers: For a given an instance, the process is equivalent to extracting marbles (the individual classifiers), without replacement, from a bag that contains a known number of marbles, but whose color (class label prediction) distribution is unknown. In addition, we show that for the specific case of a uniform prior distribution of class votes this process is equivalent to the one developed in [14]. In the current description, which does not assume a uniform distribution prior for the class votes, the hypergeometric distribution plays a central role.

The results of this statistical description are then used to design a dynamic ensemble pruning method, with the goal of speeding up predictions in the test phase. For a given instance, it is possible to compute the probability that the the partial decision made on the basis of the known votes (i.e., the class label predictions of the subset of classifiers that have been queried) and the final ensemble decision coincide. If this probability is above a specified threshold, sufficiently close to $1$, a reliable estimate of the class label that the complete ensemble would predict can be made on the basis of the known votes. The effectiveness of this dynamic ensemble pruning method is illustrated using random forests. The prior distribution of class votes is estimated using out-of-bag data. As a result of incorporating this problem-specific knowledge in the statistical analysis of the voting process, the differences between the predictions of the dynamically pruned ensemble and the complete ensemble are closer to the specified threshold than when a uniform distribution is assumed, as in SIBA [14]. In the empirical evaluation performed, this dynamic ensemble pruning algorithm consistently yields improvements of classification speed over SIBA without a significant deterioration of accuracy. Finally, the statistical model proposed is used to provide an accurate estimate of the average number of individual classifier predictions that are needed to reach a stable ensemble prediction.

**Acknowledgments**

The authors acknowledge financial support from the *Comunidad de Madrid* (project CASI-CAM-CM S2013/ICE-2845), and from the Spanish *Ministerio de Economía y Competitividad* (projects TIN2013-42351-P and TIN2015-70308-REDT).

# References

[1] A. Asuncion and D. Newman. UCI machine learning repository, 2007.

[2] J. Basilico, M. Munson, T. Kolda, K. Dixon, and W. Kegelmeyer. Comet: A recipe for learning and using large ensembles on massive data. In *Proceedings - IEEE International Conference on Data Mining, ICDM*, pages 41–50, 2011.

[3] D. Benbouzid, R. Busa-Fekete, and B. Kégl. Fast classification using sparse decision dags. In *Proceedings of the 29th International Conference on Machine Learning, ICML 2012*, volume 1, pages 951–958, 2012.

[4] L. Breiman. Bias, variance, and arcing classifiers. Technical Report 460, Statistics Department, University of California, 1996.

[5] L. Breiman. Random forests. *Machine Learning*, 45(1):5–32, 2001.

[6] R. Caruana and A. Niculescu-Mizil. Ensemble selection from libraries of models. In *Proc. of the 21st International Conference on Machine Learning (ICML'04)*, 2004.

[7] R. Caruana and A. Niculescu-Mizil. An empirical comparison of supervised learning algorithms. In *Proc. of the 23rd International Conference on Machine Learning*, pages 161–168, New York, NY, USA, 2006. ACM Press.

[8] T. G. Dietterich. Ensemble methods in machine learning. In *Multiple Classifier Systems: First International Workshop*, pages 1–15, 2000.

[9] T. G. Dietterich. An experimental comparison of three methods for constructing ensembles of decision trees: Bagging, boosting, and randomization. *Machine Learning*, 40(2):139–157, 2000.

[10] W. Fan, F. Chu, H. Wang, and P. S. Yu. Pruning and dynamic scheduling of cost-sensitive ensembles. In *Proc. of the 18th National Conference on Artificial Intelligence*, pages 146–151. American Association for Artificial Intelligence, 2002.

[11] M. Fernández-Delgado, E. Cernadas, S. Barro, and D. Amorim. Do we need hundreds of classifiers to solve real world classification problems? *Journal of Machine Learning Research*, 15:3133–3181, 2014.

[12] T. Gao and D. Koller. Active classification based on value of classifier. In *NIPS*, 2011.

[13] L. K. Hansen and P. Salamon. Neural network ensembles. *IEEE Transactions on Pattern Analysis and Machine Intelligence*, 12:993–1001, 1990.

[14] D. Hernández-Lobato, G. Martínez-Muñoz, and A. Suárez. Statistical instance-based pruning in ensembles of independent classifiers. *IEEE Transactions on Pattern Analysis and Machine Intelligence*, 31(2):364–369, 2009.

[15] T. K. Ho, J. J. Hull, and S. N. Srihari. Decision combination in multiple classifier systems. *IEEE Transactions on Pattern Analysis Machine Intelligence*, 16(1):66–75, 1994.

[16] D. D. Margineantu and T. G. Dietterich. Pruning adaptive boosting. In *Proc. of the 14th International Conference on Machine Learning*, pages 211–218. Morgan Kaufmann, 1997.

[17] F. Markatopoulou, G. Tsoumakas, and I. Vlahavas. Dynamic ensemble pruning based on multi-label classification. *Neurocomputing*, 150(PB):501–512, 2015.

[18] L. Reyzin. Boosting on a budget: Sampling for feature-efficient prediction. In L. Getoor and T. Scheffer, editors, *Proceedings of the 28th International Conference on Machine Learning (ICML-11)*, ICML '11, pages 529–536, New York, NY, USA, June 2011. ACM.

[19] R. S. Sutton, J. Modayil, M. Delp, T. Degris, P. M. Pilarski, A. White, and D. Precup. Horde: A scalable real-time architecture for learning knowledge from unsupervised sensorimotor interaction. In S. Tumer, Yolum and Stone, editors, *Proc. of 10th Int. Conf. on Autonomous Agents and Multiagent Systems (AAMAS 2011)*, pages 761–768, Taipei, Taiwan, 2011.

[20] H. Wang, W. Fan, P. S. Yu, and J. Han. Mining concept-drifting data streams using ensemble classifiers. In *KDD '03: Proceedings of the ninth ACM SIGKDD international conference on Knowledge discovery and data mining*, pages 226–235, New York, NY, USA, 2003. ACM Press.

[21] Y. Zhang, S. Burer, and W. N. Street. Ensemble pruning via semi-definite programming. *Journal of Machine Learning Research*, 7:1315–1338, 2006.

